# Clustering Sparse Graphs

**Yudong Chen**
Department of Electrical and Computer Engineering
The University of Texas at Austin
Austin, TX 78712
ydchen@utexas.edu

**Sujay Sanghavi**
Department of Electrical and Computer Engineering
The University of Texas at Austin
Austin, TX 78712
sanghavi@mail.utexas.edu

**Huan Xu**
Mechanical Engineering Department
National University of Singapore
Singapore 117575, Singapore
mpexuh@nus.edu.sg

## Abstract

We develop a new algorithm to cluster sparse unweighted graphs – i.e. partition the nodes into disjoint clusters so that there is higher density within clusters, and low across clusters. By sparsity we mean the setting where both the in-cluster and across cluster edge densities are very small, possibly vanishing in the size of the graph. Sparsity makes the problem noisier, and hence more difficult to solve.

Any clustering involves a tradeoff between minimizing two kinds of errors: missing edges within clusters and present edges across clusters. Our insight is that in the sparse case, these must be *penalized differently*. We analyze our algorithm's performance on the natural, classical and widely studied "planted partition" model (also called the stochastic block model); we show that our algorithm can cluster sparser graphs, and with smaller clusters, than all previous methods. This is seen empirically as well.

## 1  Introduction

This paper proposes a new algorithm for the following task: given a *sparse* undirected unweighted graph, partition the nodes into disjoint clusters so that the density of edges within clusters is higher than the edges across clusters. In particular, we are interested in settings where even within clusters the edge density is low, and the density across clusters is an additive (or small multiplicative) constant lower.

Several large modern datasets and graphs are sparse; examples include the web graph, social graphs of various social networks, etc. Clustering naturally arises in these settings as a means/tool for community detection, user profiling, link prediction, collaborative filtering etc. More generally, there are several clustering applications where one is given as input a set of similarity relationships, but this set is quite sparse. Unweighted sparse graph clustering corresponds to a special case in which all similarities are either "1" or "0".

As has been well-recognized, sparsity complicates clustering, because it makes the problem noisier. Just for intuition, imagine a random graph where every edge has a (potentially different) probability $p_{ij}$ (which can be reflective of an underlying clustering structure) of appearing in the graph. Consider now the edge random variable, which is 1 if there is an edge, and 0 else. Then, in the sparse graph setting of small $p_{ij} \to 0$, the mean of this variable is $p_{ij}$ but its standard deviation is $\sqrt{p_{ij}}$, which

can be much larger. This problem gets worse as $p_{ij}$ gets smaller. Another parameter governing problem difficulty is the *size* of the clusters; smaller clusters are easier to lose in the noise.

**Our contribution:** We propose a new algorithm for sparse unweighted graph clustering. Clearly, there will be two kinds of deviations (i.e. errors) between the given graph and any candidate clustering: missing edges within clusters, and present edges across clusters. Our *key realization* is that for sparse graph clustering, these two types of error should be *penalized differently*. Doing so gives as a combinatorial optimization problem; our algorithm is a particular convex relaxation of the same, based on the fact that the cluster matrix is low-rank (we elaborate below). Our **main analytical result** in this paper is theoretical guarantees on its performance for the classical *planted partition model* [10], also called the *stochastic block-model* [1, 22], for random clustered graphs. While this model has a rich literature (e.g., [4, 7, 10, 20]), we show that our algorithm *out-performs (upto at most log factors) every existing method* in this setting (i.e. it recovers the true clustering for a bigger range of sparsity and cluster sizes). Both the level of sparsity and the number and sizes of the clusters are allowed to be functions of $n$, the total number of nodes. In fact, we show that in a sense we are close to the boundary at which "any" spectral algorithm can be expected to work. Our **simulation study** confirms our theoretic finding, that the proposed method is effective in clustering sparse graphs and outperforms existing methods.

The rest of the paper is organized as follows: Section 1.1 provides an overview of related work; Section 2 presents both the precise algorithm, and the idea behind it; Section 3 presents the main results – analytical results on the planted partition / stochastic block model – which are shown to outperform existing methods; Section 4 provides simulation results; and finally, the proof of main theoretic results is outlined in Section 5.

## 1.1 Related Work

The general field of clustering, or even graph clustering, is too vast for a detailed survey here; we focus on the most related threads, and therein too primarily on work which provides theoretical "cluster recovery" guarantees on the resulting algorithms.

**Correlation clustering:** As mentioned above, every candidate clustering will have two kinds of errors; correlation clustering [2] weighs them equally, thus the objective is to find the clustering which minimizes just the total number of errors. This is an NP-hard problem, and [2] develops approximation algorithms. Subsequently, there has been much work on devising alternative approximation algorithms for both the weighted and unweighted cases, and for both agreement and disagreement objectives [12, 13, 3, 9]. Approximations based on LP relaxation [11] and SDP relaxation [25, 19], followed by rounding, have also been developed. All of this line of work is on worst-case guarantees. We emphasize that while we do convex relaxation as well, we do not do rounding; rather, our convex program itself yields an optimal clustering.

**Planted partition model / Stochastic block model:** This is a natural and classic model for studying graph clustering in the average case, and is also the setting for our performance guarantees. Our results are directly comparable to work here; we formally define this setting in section 3 and present a detailed comparison, after some notation and our theorem, in section 3 below.

**Sparse and low-rank matrix decomposition:** It has recently been shown [8, 6] that, under certain conditions, it is possible to recover a low-rank matrix from sparse errors of arbitrary magnitude; this has even been applied to graph clustering [17]. Our algorithm turns out to be a weighted version of sparse and low-rank matrix decomposition, with *different elements of the sparse part penalized differently, based on the given input.* To our knowledge, ours is the first paper to study any weighted version; in that sense, while our weights have a natural motivation in our setting, our results are likely to have broader implications, for example robust versions of PCA when not all errors are created equal, but have a corresponding prior.

## 2 Algorithm

**Idea:** Our algorithm is a convex relaxation of a natural combinatorial objective for the sparse clustering problem. We now briefly motivate this objective, and then formally describe our algorithm. Recall that we want to find a clustering (i.e. a partition of the nodes) such that in-cluster connectiv-

ity is denser than across-cluster connectivity. Said differently, we want a clustering that has a small number of *errors*, where an error is either *(a)* an edge between two nodes in different clusters, or *(b)* a missing edge between two nodes in the same cluster. A natural (combinatorial) objective is to minimize a weighted combination of the two types of errors.

The correlation clustering setup [2] gives equal weights to the two types of errors. However, for sparse graphs, this will yield clusters with a very small number of nodes. This is because there is sparsity both within clusters and across clusters; grouping nodes in the same cluster will result in a lot of errors of type (b) above, without yielding corresponding gains in errors of type (a) – *even when* they may actually be in the same cluster. This can be very easily seen: suppose, for example, the "true" clustering has two clusters with equal size, and the in-cluster and across-cluster edge density are both less than 1/4. Then, when both errors are weighted equally, the clustering which puts every node in a separate cluster will have lower cost than the true clustering.

To get more meaningful solutions, we *penalize the two types of errors differently.* In particular, sparsity means that we can expect many more errors of type (b) in any solution, and hence we should give this (potentially much) smaller weight than errors of type (a). Our *crucial insight* is that we can know what kind of error will (potentially) occur on any given edge *from the given adjacency matrix itself.* In particular, if $a_{ij} = 1$ for some pair $i, j$, when in any clustering it will either have no error, or an error of type (a); it will never be an error of type (b). Similarly if $a_{ij} = 0$ then it can only be an error of type (b), if at all. Our algorithm is a convex relaxation of the combinatorial problem of finding the minimum cost clustering, with the *cost for an error on edge $i, j$ determined based on the value of $a_{ij}$.* Perhaps surprisingly, this simple idea yields better results than the extensive literature already in place for planted partitions.

We proceed by representing the given adjacency matrix $A$ as the sum of two matrices $A = Y + S$, where we would like $Y$ to be a *cluster matrix*, with $y_{ij} = 1$ if and only if $i, j$ are in the same cluster, and 0 otherwise[1][2]. $S$ is the corresponding error matrix as compared to the given $A$, and has values of +1, -1 and 0.

We now make a cost matrix $C \in \mathbb{R}^{n \times n}$ based on the insight above; we choose two values $c_{\mathcal{A}}$ and $c_{\mathcal{A}^c}$ and set $c_{ij} = c_{\mathcal{A}}$ if the corresponding $a_{ij} = 1$, and $c_{ij} = c_{\mathcal{A}^c}$ if $a_{ij} = 0$. However, diagonal $c_{ii} = 0$. With this setup, we have

$$\textbf{Combinatorial Objective:} \qquad \min_{Y,S} \quad \|C \circ S\|_1 \qquad\qquad (1)$$
$$s.t \quad Y + S = A$$
$$Y \text{ is a cluster matrix}$$

Here $C \circ S$ denotes the matrix obtained via element-wise product between the two matrices $C, S$, i.e. $(C \circ S)_{ij} = c_{ij} s_{ij}$. Also $\| \cdot \|_1$ denotes the element-wise $\ell_1$ norm (i.e. sum of absolute values of elements).

**Algorithm:** Our algorithm involves solving a convex relaxation of this combinatorial objective, by replacing the "$Y$ is a cluster matrix" constraint with *(i)* constraints $0 \leq y_{ij} \leq 1$ for all elements $i, j$, and *(ii)* a nuclear norm[3] penalty $\|Y\|_*$ in the objective. The latter encourages $Y$ to be *low-rank*, and is based on the well-established insight that the cluster matrix (being a block-diagonal collection of 1's) is low-rank. Thus we have our algorithm:

$$\textbf{Sparse Graph Clustering:} \qquad \min_{Y,S} \quad \|Y\|_* + \|C \circ S\|_1 \qquad\qquad (2)$$
$$s.t. \quad 0 \leq y_{ij} \leq 1, \forall i, j \qquad\qquad (3)$$
$$Y + S = A,$$

Once $\widehat{Y}$ is obtained, check if it is a cluster matrix (say e.g. via an SVD, which will also reveal cluster membership if it is). If it is not, any one of several rounding/aggregration ideas can be used empirically. Our theoretical results provide sufficient conditions under which the optimum of the convex program is integral and a clustering, with no rounding required. Section 3 in the supplementary material provides details on fast implementation for large matrices; this is one reason

we did not include a semidefinite constraint on $Y$ in our algorithm. Our algorithm has two positive parameters: $c_{\mathcal{A}}, c_{\mathcal{A}^c}$. We defer discussion on how to choose them until after our main result.

**Comments:** Based on the given $A$ and these values, the optimal $\widehat{Y}$ may or may not be a cluster matrix. If $\widehat{Y}$ is a cluster matrix, then clearly it minimizes the combinatorial objective above. Additionally, it is not hard to see (proof in the supplementary material) that its performance is "monotone", in the sense that adding edges "aligned with" $\widehat{Y}$ cannot result in a different optimum, as summarized in the following lemma. This shows that, in the terminology of [19, 4, 14], our method is robust under a classical semi-random model where an adversary can add edge within clusters and remove edges between clusters.

**Lemma 1.** *Suppose $\widehat{Y}$ is the optimum of Formulation (2) for a given A. Suppose now we arbitrarily change some edges of A to obtain $\widetilde{A}$, by (a) choosing some edges such that $\widehat{y}_{ij} = 1$ but $a_{ij} = 0$, and making $\widetilde{a}_{ij} = 1$, and (b) choosing some edges where $\widehat{y}_{ij} = 0$ but $a_{ij} = 1$, and making $\widetilde{a}_{i}j = 0$. Then, $\widehat{Y}$ is also an optimum of Formulation (2) with $\widetilde{A}$ as the input.*

Our theoretical guarantees characterize when the optimal $\widehat{Y}$ will be a cluster matrix, and recover the clustering, in a natural classical problem setting called the planted partition model [10]. These theoretical guarantees also provide guidance on how one would pick parameter values in practice; we thus defer discussion on parameter picking until after we present our main theorem.

## 3 Performance Guarantees

In this section we provide analytical performance guarantees for our algorithm under a natural and classical graph clustering setting: (a generalization of) the planted partition model [10]. We first describe the model, and then our results.

**(Generalized) Planted partition model:** Consider a random graph generated as follows: the $n$ nodes are partitioned into $r$ disjoint clusters, which we will refer to as the "true" clusters. Let $K$ be the *minimum cluster size*. For every pair of nodes $i, j$ that belong to the same cluster, edge $(i, j)$ is present in the graph with probability that is *at least* $\bar{p}$, while for every pair where the nodes are in different clusters the edge is present with probability *at most* $\bar{q}$. We call this model the "generalized" planted partition because we allow for clusters to be different sizes, and the edge probabilities also to be different (but uniformly bounded as mentioned). The **objective** is to find the partition, given the random graph generated from it.

Recall that $A$ is the given adjacency matrix, and let $Y^*$ be the matrix corresponding to the true clusters as above – i.e. $y_{ij}^* = 1$ if and only if $i, j$ are in the same true cluster, and 0 otherwise.. Our result below establishes conditions under which our algorithm, specifically the convex program (2)-(3), yields this $Y^*$ as the unique optimum (without any further need for rounding etc.) with high probability (w.h.p.). Throughout the paper, *with high probability* means with probability at least $1 - c_0 n^{-10}$ for some absolute constant $c_0$

**Theorem 1.** *Suppose we choose $c_{\mathcal{A}} = \frac{1}{16\sqrt{n \log n}} \min \left\{ \sqrt{\frac{1-\bar{q}}{\bar{q}}}, \sqrt{\frac{n}{\log^4 n}} \right\}$, and $c_{\mathcal{A}^c} = \frac{1}{16\sqrt{n \log n}} \min \left\{ \sqrt{\frac{\bar{p}}{1-\bar{p}}}, 1 \right\}$. Then $(Y^*, A - Y^*)$ is the unique optimal solution to Formulation (2) w.h.p. provided $\bar{q} \leq \frac{1}{4}$, and*

$$\frac{\bar{p} - \bar{q}}{\sqrt{\bar{p}}} \geq c_1 \frac{\sqrt{n}}{K} \log^2 n.$$

*where $c_1$ is an absolute positive constant.*

Our theorem quantifies the tradeoff between the two quantities governing the hardness of a planted partition problem – the difference in edge densities $p-q$, and the minimum cluster size $K$ – required for our algorithm to succeed, i.e. to recover the planted partition *without any error*. Note that here $p, q$ and $K$ are allowed to scale with $n$. We now discuss and remark on our result, and then compare its performance to past approaches and theoretical results in Table 1.

Note that we need $K$ to be $\Omega(\sqrt{n} \log^2 n)$. This will be achieved only when $\bar{p} - \bar{q}$ is a constant that does not change with $n$; indeed in this extreme our theorem becomes a "dense graph" result,

matching e.g. the scaling in [17, 19]. If $\frac{\bar{p}-\bar{q}}{\sqrt{\bar{p}}}$ decreases with $n$, corresponding to a sparser regime, then the minimum size of $K$ required will increase.

A nice feature of our work is that we only need $\bar{p}-\bar{q}$ to be large *only as compared to* $\sqrt{\bar{p}}$; several other existing results (see Table 1) require a lower bound (as a function only of $n$, or $n,K$) on $\bar{p}-\bar{q}$ itself. This allows us to guarantee recovery for *much sparser graphs* than all existing results. For example, when $K$ is $\Theta(n)$, $\bar{p}$ and $\bar{p}-\bar{q}$ can be as small as $\Theta(\frac{\log^4 n}{n})$. This scaling is close to optimal: if $\bar{p} < \frac{\log n}{n}$ then each cluster will be almost surely disconnected, and if $\bar{p}-\bar{q} = o(\frac{1}{n})$, then on average a node has equally many neighbours in its own cluster and in another cluster – both are ill-posed situations in which one can not hope to recover the underlying clustering. When $K = \Omega\left(\sqrt{n}\log^2 n\right)$, $\bar{p}$ and $\bar{p}-\bar{q}$ can be $\Theta\left(\frac{n\log^4 n}{K^2}\right)$, while the previous best result for this regime requires at least $\Theta\left(\frac{n^2}{K^3}\right)$ [20].

**Parameters:** Our algorithm has two parameters: $c_\mathcal{A}$ and $c_{\mathcal{A}^c}$. The theorem provides a way to choose their values, *assuming* we know the values of the bounds $\bar{p},\bar{q}$. To estimate these from data, we can use the following rule of thumb; our empirical results are based on this rule. If all the clusters have equal size $K$, it is easy to verify that the first eigenvalue of $\mathbb{E}[A-I]$ is $K(p-q)-p+nq$ with multiplicity 1, the second eigenvalue is $K(p-q)-p$ with multiplicity $\frac{n}{K}-1$, and the third eigenvalue is $-p$ with multiplicities $(n-\frac{n}{K})$ [16]. We thus have the following rule of thumb:

1. Compute the eigenvalues of $A-I$, denoted as $\lambda_1,\ldots,\lambda_n$.

2. Let $r = \arg\max_{i=1,\ldots,n-1}(\lambda_i - \lambda_{i-1})$. Set $K = n/r$.

3. Solve for $p$ and $q$ from the equations

$$\begin{cases} K(p-q)-p+nq = \lambda_1 \\ K(p-q)-p = \lambda_2 \end{cases}$$

Table 1: **Comparison with literature.** This table shows the lower-bound requirements on $K$ and $p-q$ that existing literature needs for exact recovery of the planted partitions/clusters. Note that this table is under the assumption that every cluster is of size $K$, and the edge densities are uniformly $p$ and $q$ (for within and across clusters respectively). As can be seen, our algorithm achieves a better $p-q$ scaling than every other result. And, we achieve a better $K$ scaling than every other result except Shamir [23], Oymak & Hassibi [21] and Giesen & Mitsche[15]; we are off by a at most $\log^2 n$ factor from each of these. Perhaps more importantly, we use a completely different algorithmic approach from all of the others.

| Paper | Min. cluster size $K$ | Density difference $p-q$ |
|---|---|---|
| Boppana [5] | $n/2$ | $\Omega(\frac{\sqrt{p}\log n}{\sqrt{n}})$ |
| Jerrum & Sorkin [18] | $n/2$ | $\Omega(\frac{1}{n^{1/6-\epsilon}})$ |
| Condon & Karp [10] | $\Omega(n)$ | $\Omega(\frac{1}{n^{1/2-\epsilon}})$ |
| Carson & Impaglizzo [7] | $n/2$ | $\omega(\frac{\sqrt{p}}{\sqrt{n}}\log n)$ |
| Feige & Kilian [14] | $n/2$ | $\Omega(\frac{1}{n}\log n)$ |
| Shamir [23] | $\Omega(\sqrt{n}\log n)$ | $\Omega(\frac{\sqrt{n}\log n}{K})$ |
| McSherry [20] | $\Omega(n^{2/3})$ | $\Omega(\sqrt{\frac{pn^2}{K^3}})$ |
| Oymak & Hassibi [21] | $\Omega(\sqrt{n})$ | $\Omega(\max\{\frac{\sqrt{n}}{K},\sqrt{\frac{\log n}{K}}\})$ |
| Giesen & Mitsche[15] | $\Omega(\sqrt{n})$ | $\Omega(\frac{\sqrt{n}}{K})$ |
| Bollobas [4] | $\Omega(\frac{n}{\log^{1/8} n})$ | $\Omega(\max\{\sqrt{\frac{q\log n}{n}},\frac{\log n}{n}\})$ |
| | | |
| This paper | $\Omega(\sqrt{n}\log^2 n)$ | $\Omega(\frac{\sqrt{pn}\log^2 n}{K})$ |

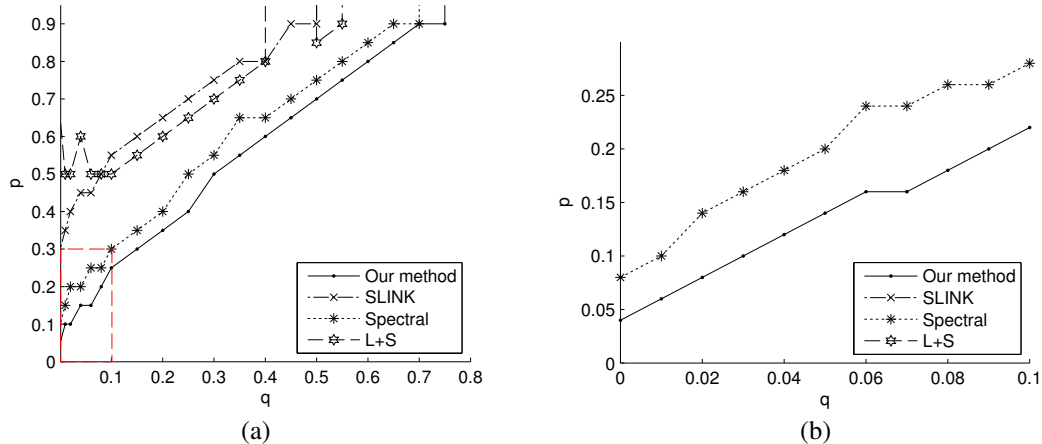

Figure 1: (a) Comparison of our method with Single-Linkage clustering (SLINK), spectral cluster-
ing, and low-rank-plus-sparse (L+S) approach. The area above each curve is the values of $(p, q)$ for
which a method successfully recovers the underlying true clustering. (b) More detailed results for
the area in the box in (a). The experiments are conducted on synthetic data with $n = 1000$ nodes
and $r = 5$ clusters with equal size $K = 200$.

## 4    Empirical Results

We perform experiments on synthetic data, and compare with other methods. We generate a graph
using the planted partition model with $n = 1000$ nodes, $r = 5$ clusters with equal size $K = 200$,
and $p, q \in [0, 1]$. We apply our method to the data, where we use the fast solver described in the
supplementary material. We estimate $p$ and $q$ using the heuristic described in Section 3, and choose
the weights $c_\mathcal{A}$ and $c_{\mathcal{A}^c}$ according to the main theorem[4]. Due to numerical accuracy, the output $\hat{Y}$
of our algorithm may not be integer, so we do the following simple rounding: compute the mean
$\bar{y}$ of the entries of $\hat{Y}$, and round each entry of $\hat{Y}$ to 1 if it is greater than $\bar{y}$, and 0 otherwise. We
measure the error by $\|Y^* - \text{round}(\hat{Y})\|_1$, which is simply the number of misclassifed pairs. We say
our method succeeds if it misclassifies less than $0.1\%$ of the pairs.

For comparison, we consider three alternative methods: (1) Single-Linkage clustering (SLINK) [24],
which is a hierarchical clustering method that merge the most similar clusters in each iteration. We
use the difference of neighbours, namely $\|A_{i\cdot} - A_{j\cdot}\|_1$, as the distance measure of node $i$ and $j$, and
output when SLINK finds a clustering with $r = 5$ clusters. (2) A spectral clustering method [26],
where we run SLINK on the top $r = 5$ singular vectors of $A$. (3) Low-rank-plus-sparse approach
[17, 21], followed by the same rounding scheme. Note the first two methods assume knowledge of
$r$, which is not available to our method. Success is measured in the same way as above.

For each $q$, we find the smallest $p$ for which a method succeeds, and average over 20 trials. The
results are shown in Figure 1(a), where the area above each curves corresponds to the range of
feasible $(p, q)$ for each method. It can been seen that our method subsumes all others, in that we
succeed for a strictly larger range of $(p, q)$. Figure 1(b) shows more detailed results for sparse graphs
$(p \le 0.3, q \le 0.1)$, for which SLINK and trace-norm-plus unweighted $\ell_1$ completely fail, while our
method significantly outperforms the spectral method, the only alternative method that works in this
regime.

## 5    Proof of Theorem 1

**Overview:** Let $S^* \triangleq A - Y^*$. The proof consists of two main steps: *(a)* developing a new approxi-
mate dual certificate condition, i.e. a set of stipulations which, if satisfied by any matrix $W$, would

performance, which we do not pursue here

guarantee the optimality of $(Y^*, S^*)$, and *(b)* constructing a $W$ that satisfies these stipulations with high probability. While at a high level these two steps have been employed in several papers on sparse and low-rank matrix decomposition, our analysis is different because it relies critically on the specific clustering setting we are in. Thus, even though we are looking at a potentially more involved setting with input-dependent weights on the sparse matrix regularizer, our proof is much simpler than several others in this space. Also, existing proofs do not cover our setting.

**Preliminaries:** Define support sets $\Omega \triangleq \mathrm{support}(S^*)$, and $R \triangleq \mathrm{support}(Y^*)$. Their complements are $\Omega^c$ and $R^c$ respectively. Due to the constraints (3) in our convex program, if $(Y^* + \Delta, S^* - \Delta)$ is a feasible solution to the convex program (2), then it has to be that $\Delta \in \mathfrak{D}$, where

$$\mathfrak{D} \triangleq \{M \in \mathbb{R}^{n \times n} \mid \forall (i,j) \in R: \ -1 \le m_{ij} \le 0; \quad \forall (i,j) \in R^c: \ 1 \ge m_{ij} \ge 0\}.$$

Thus we only need to execute steps *(a),(b)* above for optimality over this restricted set of deviations. Finally, we define the (now standard) projection operators: $P_\Omega(M)$ is the matrix where the $(i,j)^{th}$ entry is $m_{ij}$ if $(i,j) \in \Omega$, and 0 else. Let the SVD of $Y^*$ be $U_0 \Sigma_0 U_0^\top$ (notice that $Y^*$ is a symmetric positive semidefinite matrix), and let $P_{T^\perp}(M) \triangleq (I - U_0 U_0^\top) M (I - U_0 U_0^\top)$ be the projection of $M$ onto the space of matrices whose columns and rows are orthogonal to those of $Y^*$, and $P_T(M) \triangleq M - P_{T^\perp}(M)$.

**Step $(a)$ - Dual certificate condition:** The following proposition provides a sufficient condition for the optimality of $(Y^*, S^*)$.

**Proposition 1 (New Dual Certificate Conditions for Clustering).** *If there exists a matrix $W \in \mathbb{R}^{n \times n}$ and a positive number $\epsilon$ obeying the following conditions*

1. $\|P_{T^\perp} W\| \le 1$.

2. $\|P_T(W)\|_\infty \le \frac{\epsilon}{2} \min \{c_{\mathcal{A}^c}, c_{\mathcal{A}}\}$

3. $\left\langle P_\Omega(U_0 U_0^\top + W), \Delta \right\rangle = (1 + \epsilon) \|P_\Omega(C \circ \Delta)\|_1, \forall \Delta \in \mathfrak{D}.$

4. $\left\langle P_{\Omega^c}(U_0 U_0^\top + W), \Delta \right\rangle \ge -(1 - \epsilon) \|P_{\Omega^c}(C \circ \Delta)\|_1, \forall \Delta \in \mathfrak{D}$

*then $(Y^*, S^*)$ is the unique optimal solution to the convex program* (2).

The proof is in the supplementary material; it also involves several steps unique to our clustering setup here.

**Step $(b)$ - Dual certificate constructions:** We now construct a $W$, and show that it satisfies the conditions in Proposition 1 w.h.p. (but *not always*, and this is key to its simple construction). To keep the notation light, we consider the standard planted partition model, where the edge probabilities are uniform; that is, for every pair of nodes in the same cluster, there is an edge between them with probability $p \ge \bar{p}$, and for every pair where the nodes are in different clusters, the edge is present with probability $q \le \bar{q}$. It is straightforward to adapt the proof to the general case with non-uniform edge probabilities. We define $W \triangleq W_1 + W_2$ where

$$W_1 \ \triangleq \ -P_\Omega(U_0 U_0^\top) + \sum_{m=1}^r \frac{1-p}{p} \frac{1}{k_m} \mathbf{1}_{R_m \cap \Omega^c},$$

$$W_2 \ \triangleq \ (1 + \epsilon) \left[ C \circ S^* + \frac{c_{\mathcal{A}^c}(1-p)}{p} \mathbf{1}_{R \cap \Omega^c} - \frac{c_{\mathcal{A}} q}{1-q} \mathbf{1}_{R^c \cap \Omega^c} \right].$$

Intuitively speaking, the idea is that $W_1$ and $W_2$ are zero mean random matrices, so they are likely to have small norms. To prove Theorem 1, it remains to show that $W$ satisfies the desired conditions w.h.p.; this is done below, with proof in the supplementary, and is much simpler than similar proofs in the sparse-plus-low-rank literature.

**Proposition 2.** *Under the assumptions of Theorem 1, with high probability, $W$ satisfies the conditions in Proposition 1 with $\epsilon = \frac{2 \log^2 n}{K} \sqrt{\frac{n}{\bar{p}}}$.*

# 6  Conclusion

We presented a convex optimization formulation, essentially a weighted version of low-rank matrix decomposition, to address graph clustering where the graph is *sparse*. We showed that under a wide range of problem parameters, the proposed method guarantees to recover the correct clustering. In fact, our theoretic analysis shows that the proposed method outperforms, i.e., succeeds under less restrictive conditions, every existing method in this setting. Simulation studies also validates the efficiency and effectiveness of the proposed method.

This work is motivated by analyzing large-scale social network, where inherently, even actors (nodes) within one cluster are more than likely not having connections. As such, immediate goals for future work include faster algorithm implementations, as well as developing effective postprocessing schemes (e.g., rounding) when the obtained solution is not an exact cluster matrix.

### Acknowledgments

S. Sanghavi would like to acknowledge NSF grants 0954059 and 1017525, and ARO grant W911NF1110265. The research of H. Xu is partially supported by the Ministry of Education of Singapore through NUS startup grant R-265-000-384-133.

## Footnotes

[1] In this paper we will assume the convention that $a_{ii} = 1$ and $y_{ii} = 1$ for all nodes $i$.

[2] In other words, $Y$ is the adjacency matrix of a graph consisting of disjoint cliques.

[3] The nuclear norm of a matrix is the sum of its singular values.

[4]we point out that searching for the best $c_\mathcal{A}$ and $c_{\mathcal{A}^c}$ while keeping $c_\mathcal{A}/c_{\mathcal{A}^c}$ fixed might lead to better

# References

[1] P. Holland andK.B. Laskey and S. Leinhardt. Stochastic blockmodels: Some first steps. *Social Networks*, 5:109–137, 1983.

[2] N. Bansal, A. Blum, and S. Chawla. Correlation clustering. *Machine Learning*, 56(1):89–113, 2004.

[3] H. Becker. A survey of correlation clustering. Available online at http://www1.cs.columbia.edu/ hila/clustering.pdf, 2005.

[4] B. Bollobás and AD Scott. Max cut for random graphs with a planted partition. *Combinatorics, Probability and Computing*, 13(4-5):451–474, 2004.

[5] R.B. Boppana. Eigenvalues and graph bisection: An average-case analysis. In *Foundations of Computer Science, 1987., 28th Annual Symposium on*, pages 280–285. IEEE, 1987.

[6] E.J. Candes, X. Li, Y. Ma, and J. Wright. Robust principal component analysis? *Arxiv preprint arXiv:0912.3599*, 2009.

[7] T. Carson and R. Impagliazzo. Hill-climbing finds random planted bisections. In *Proceedings of the twelfth annual ACM-SIAM symposium on Discrete algorithms*, pages 903–909. Society for Industrial and Applied Mathematics, 2001.

[8] V. Chandrasekaran, S. Sanghavi, S. Parrilo, and A. Willsky. Rank-sparsity incoherence for matrix decomposition. *SIAM Journal on Optimization*, 21(2):572–596, 2011.

[9] M. Charikar, V. Guruswami, and A. Wirth. Clustering with qualitative information. In *Foundations of Computer Science, 2003. Proceedings. 44th Annual IEEE Symposium on*, pages 524–533. IEEE, 2003.

[10] A. Condon and R.M. Karp. Algorithms for graph partitioning on the planted partition model. *Random Structures and Algorithms*, 18(2):116–140, 2001.

[11] E. Demaine and N. Immorlica. Correlation clustering with partial information. *Approximation, Randomization, and Combinatorial Optimization.. Algorithms and Techniques*, pages 71–80, 2003.

[12] E.D. Demaine, D. Emanuel, A. Fiat, and N. Immorlica. Correlation clustering in general weighted graphs. *Theoretical Computer Science*, 361(2):172–187, 2006.

[13] D. Emanuel and A. Fiat. Correlation clustering–minimizing disagreements on arbitrary weighted graphs. *Algorithms-ESA 2003*, pages 208–220, 2003.

[14] U. Feige and J. Kilian. Heuristics for semirandom graph problems. *Journal of Computer and System Sciences*, 63(4):639–671, 2001.

[15] J. Giesen and D. Mitsche. Bounding the misclassification error in spectral partitioning in the planted partition model. In *Graph-Theoretic Concepts in Computer Science*, pages 409–420. Springer, 2005.

[16] J. Giesen and D. Mitsche. Reconstructing many partitions using spectral techniques. In *Fundamentals of Computation Theory*, pages 433–444. Springer, 2005.

[17] A. Jalali, Y. Chen, S. Sanghavi, and H. Xu. Clustering partially observed graphs via convex optimization. *Arxiv preprint arXiv:1104.4803*, 2011.

[18] M. Jerrum and G.B. Sorkin. The metropolis algorithm for graph bisection. *Discrete Applied Mathematics*, 82(1-3):155–175, 1998.

[19] C. Mathieu and W. Schudy. Correlation clustering with noisy input. In *Proceedings of the Twenty-First Annual ACM-SIAM Symposium on Discrete Algorithms*, pages 712–728. Society for Industrial and Applied Mathematics, 2010.

[20] F. McSherry. Spectral partitioning of random graphs. In *Foundations of Computer Science, 2001. Proceedings. 42nd IEEE Symposium on*, pages 529–537. IEEE, 2001.

[21] S. Oymak and B. Hassibi. Finding dense clusters via "low rank+ sparse" decomposition. *Arxiv preprint arXiv:1104.5186*, 2011.

[22] K. Rohe, S. Chatterjee, and B. Yu. Spectral clustering and the high-dimensional stochastic block model. Technical report, Technical Report 791, Statistics Department, UC Berkeley, 2010.

[23] R. Shamir and D. Tsur. Improved algorithms for the random cluster graph model. *Random Structures & Algorithms*, 31(4):418–449, 2007.

[24] R. Sibson. Slink: an optimally efficient algorithm for the single-link cluster method. *The Computer Journal*, 16(1):30–34, 1973.

[25] C. Swamy. Correlation clustering: maximizing agreements via semidefinite programming. In *Proceedings of the fifteenth annual ACM-SIAM symposium on Discrete algorithms*, pages 526–527. Society for Industrial and Applied Mathematics, 2004.

[26] U. Von Luxburg. A tutorial on spectral clustering. *Statistics and Computing*, 17(4):395–416, 2007.

